# Ensemble Methods for Phoneme Classification

**Steve Waterhouse**          **Gary Cook**
Cambridge University Engineering Department
Cambridge CB2 1PZ, England, Tel: [+44] 1223 332754
Email: srw1001@eng.cam.ac.uk, gdc@eng.cam.ac.uk

## Abstract

This paper investigates a number of ensemble methods for improving the performance of phoneme classification for use in a speech recognition system. Two ensemble methods are described; boosting and mixtures of experts, both in isolation and in combination. Results are presented on two speech recognition databases: an isolated word database and a large vocabulary continuous speech database. These results show that principled ensemble methods such as boosting and mixtures provide superior performance to more naive ensemble methods such as averaging.

## INTRODUCTION

There is now considerable interest in using ensembles or committees of learning machines to improve the performance of the system over that of a single learning machine. In most neural network ensembles, the ensemble members are trained on either the same data (Hansen & Salamon 1990) or different subsets of the data (Perrone & Cooper 1993). The ensemble members typically have different initial conditions and/or different architectures. The subsets of the data may be chosen at random, with prior knowledge or by some principled approach e.g. clustering. Additionally, the outputs of the networks may be combined by any function which results in an output that is consistent with the form of the problem. The expectation of ensemble methods is that the member networks pick out different properties present in the data, thus improving the performance when their outputs are combined.

The two techniques described here, boosting (Drucker, Schapire & Simard 1993) and mixtures of experts (Jacobs, Jordan, Nowlan & Hinton 1991), differ from simple ensemble methods.

In boosting, each member of the ensemble is trained on patterns that have been filtered by previously trained members of the ensemble. In mixtures, the members of the ensemble, or "experts", are trained on data that is stochastically selected by a gate which additionally learns how to best combine the outputs of the experts.

The aim of the work presented here is twofold and inspired from two differing but complimentary directions. Firstly, how does one select which data to train the ensemble members on and secondly, given these members how does one combine them to achieve the optimal result? The rest of the paper describes how a combination of boosting and mixtures may be used to improve phoneme error rates.

## PHONEME CLASSIFICATION

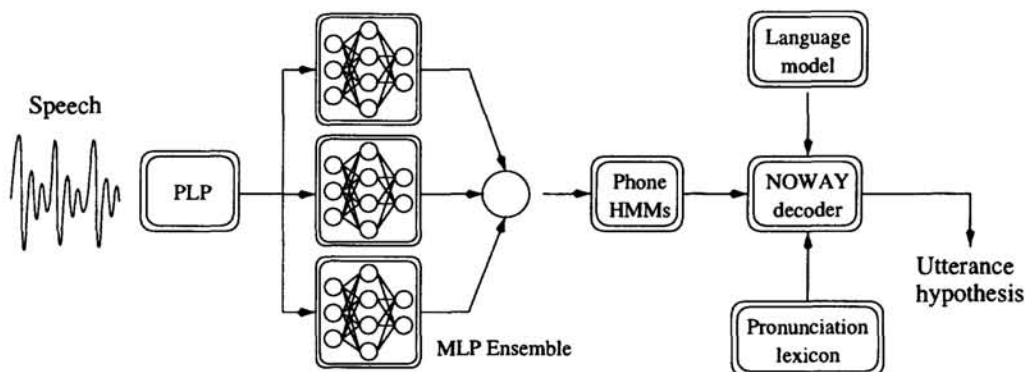

Figure 1: The ABBOT hybrid connectionist-HMM speech recognition system with an MLP ensemble acoustic model

The Cambridge University Engineering Department connectionist speech recognition system (ABBOT) uses a hybrid connectionist - hidden Markov model (HMM) approach. This is shown in figure 1. A connectionist acoustic model is used to map each frame of acoustic data to posterior phone probabilities. These estimated phone probabilities are then used as estimates of the observation probabilities in an HMM framework. Given new acoustic data and the connectionist-HMM framework, the maximum a posteriori word sequence is then extracted using a single pass, start synchronous decoder. A more complete description of the system can be found in (Hochberg, Renals & Robinson 1994).

Previous work has shown how a novel boosting procedure based on utterance selection can be used to increase the performance of the recurrent network acoustic model (Cook & Robinson 1996). In this work a combined boosting and mixtures-of-experts approach is used to improve the performance of MLP acoustic models. Results are presented for two speech recognition tasks. The first is phonetic classification on a small isolated digit database. The second is a large vocabulary continuous speech recognition task from the Wall Street Journal corpus.

## ENSEMBLE METHODS

Most ensemble methods can be divided into two separate methods; network selection and network combination. Network selection addresses the question of how to

choose the data each network is trained on. Network combination addresses the question of what is the best way to combine the outputs of these trained networks. The simplest method for network selection is to train separate networks on separate regions of the data, chosen either randomly, with prior knowledge or according to some other criteria, e.g. clustering.

The simplest method of combining the outputs of several networks is to form an average, or simple additive merge: $y(t) = \frac{1}{K} \sum_{k=1}^{K} y_k(t)$, where $y_k(t)$ is the output of the $k^{th}$ network at time $t$.

## Boosting

Boosting is a procedure which results in an ensemble of networks. The networks in a boosting ensemble are trained sequentially on data that has been filtered by the previously trained networks in the ensemble. This has the advantage that only data that is likely to result in improved generalization performance is used for training. The first practical application of a boosting procedure was for the optical character recognition task (Drucker et al. 1993). An ensemble of feedforward neural networks was trained using supervised learning. Using boosting the authors reported a reduction in error rate on ZIP codes from the United States Postal Service of 28% compared to a single network. The boosting procedure is as follows: train a network on a randomly chosen subset of the available training data. This network is then used to filter the remaining training data to produce a training set for a second network with an even distribution of cases which the first network classifies correctly and incorrectly. After training the second network the first and second networks are used to produce a training set for a third network. This training set is produced from cases in the remaining training data that the first two networks disagree on.

The boosted networks are combined using either a voting scheme or a simple add as described in the previous section. The voting scheme works as follows: if the first two networks agree, take their answer as the output, if they disagree, use the third network's answer as the output.

## Mixtures of Experts

The mixture of experts (Jacobs et al. 1991) is a different type of ensemble to the two considered so far. The ensemble members or *experts* are trained with data which is stochastically selected by a *gate*. The gate in turn learns how to best combine the experts given the data. The training of the experts, which are typically single or multi-layer networks, proceeds as for standard networks, with an additional weighting of the output error terms by the posterior probabilty $h_i(t)$ of selecting expert $i$ given the current data point at time (t): $h_i(t) = g_i(t).P_i(t) \big/ \sum_j g_j(t).P_j(t)$, where $g_i(t)$ is the output of the gate for expert $i$, and $P_i(t)$ is the probability of obtaining the correct output given expert $i$. In the case of classification, considered here, the experts use softmax output units. The gate, which is typically a single or multi-layered network with softmax output units is trained using the posterior probabilities as targets. The overall output $y(t)$ of the mixture of experts is given by the weighted combination of the gate and expert outputs: $y(t) = \sum_{k=1}^{K} g_k(t).y_k(t)$, where $y_k(t)$ is the output of the $k^{th}$ expert, and $g_k(t)$ is the output of the gate for

expert $k$ at time $t$.

The mixture of experts is based on the principle of divide and conquer, in which a relatively hard problem is broken up into a series of smaller easier to solve problems. By using the posterior probabilities to weight the experts and provide targets for the gate, the effective data sets used to train each expert may overlap.

## SPEECH RECOGNITION RESULTS

This section describes the results of experiments on two speech databases: the Bellcore isolated digits database and the Wall Street Journal Corpus (Paul & Baker 1992). The inputs to the networks consist of 9 frames of acoustic feature vectors; the frame on which the network is currently performing classification, plus 4 frames of left context and 4 frames of right context. The context frames allow the network to take account of the dynamical nature of speech. Each acoustic feature vector consists of 8th order PLP plus log energy coefficients along with the dynamic delta coeficients of these coefficients, computed with an analysis window of 25ms, every 12.5 ms at a sampling rate of 8kHz. The speech is labelled with 54 phonemes according to the standard ABBOT phone set.

### Bellcore Digits

The Bellcore digits database consists of 150 speakers saying the words "zero" through "nine", "oh", "no" and "yes". The database was divided into a training set of 122 speakers, a cross validation set of 13 speakers and a test set of 15 speakers. Each method was evaluated over 10 partitions of the data into different training, cross validation and test sets. In all the experiments on the Bellcore digits multi-layer perceptrons with 200 hidden units were used as the basic network members in the ensembles. The gates in the mixtures were also multi-layer perceptrons with 20 hidden units.

| Ensemble | Combination Method | Phone Error Rate | |
|---|---|---|---|
| | | Average | $\sigma$ |
| Simple ensemble | cheat | 14.7 % | 0.9 |
| Simple ensemble | vote | 20.3 % | 1.2 |
| Simple ensemble | average | 19.3 % | 1.2 |
| Simple ensemble | soft gated | 20.9 % | 1.2 |
| Simple ensemble | hard gated | 19.3 % | 1.0 |
| Simple ensemble | mixed | 17.1 % | 1.3 |
| Boosted ensemble | cheat | 11.9 % | 1.0 |
| Boosted ensemble | vote | 17.8 % | 1.1 |
| Boosted ensemble | average | 17.4 % | 1.1 |
| Boosted ensemble | soft gated | 17.8 % | 1.0 |
| Boosted ensemble | hard gated | 17.4 % | 1.2 |
| Boosted ensemble | mixed | 16.4 % | 1.0 |

Table 1: Comparison of phone error rates using different ensemble methods on the Bellcore isolated digits task.

Table 1 summarises the results obtained on the Bellcore digits database. The meaning of the entries are as follows. Two types of ensemble were trained:

*Simple Ensemble*: consisting of 3 networks each trained on 1/3 of the training data each (corresponding to 40 speakers used for training and 5 for cross validation for each network),

*Boosted Ensemble*: consisting of 3 networks trained according to the boosting algorithm of the previous section. Due to the relatively small size of the data set, it was necessary to ensure that the distributions of the randomly chosen data were consistent with the overall training data distribution.

Given each set of ensemble networks, 6 combination methods were evaluated:

*cheat*: The cheat scheme uses the best ensemble member for each example in the data set. The best ensemble member is determined by looking at the correct label in the labelled test set (hence *cheating*). This method is included as a lower bound on the error. Since the tests are performed on unseen data, this bound can only be approached by learning an appropriate combination function of the ensemble member outputs.

*average*: The ensemble members' outputs are combined using a simple average.

*vote*: The voting scheme outlined in the previous section.

*gated*: In the gated combination method, the ensemble networks were kept fixed whilst the gate was trained. Two types of gating were evaluated, standard or *soft* gating, and *hard* or winner take all (WTA) training. In WTA training the targets for the gate are binary, with a target of 1.0 for the output corresponding to the expert whose probability of generating the current data point correctly is greatest, and 0.0 for the other outputs.

*mixed*: In contrast to the *gated* method, the *mixed* combination method both trains a gate and retrains the ensemble members using the mixture of experts framework.

From these results it can be concluded that boosting provides a significant improvement in performance over a simple ensemble. In addition, by training a gate to combine the boosted networks performance can be further enhanced. As might be expected, re-training both the boosted networks and the gate provides the biggest improvement, as shown by the result for the mixed boosted networks.

**Wall Street Journal Corpus**

The training data used in these experiments is the short term speakers from the Wall Street Journal corpus. This consists of approximately 36,400 sentences from 284 different speakers (SI284). The first network is trained on 1.5 million frames randomly selected from the available training data (15 million frames). This is then used to filter the unseen training data to select frames for training the second network. The first and second networks are then used to select data for the third network as described previously. The performance of the boosted MLP ensemble

| Test Set | Language Model | Lexicon | Word Error Rate | | | |
|---|---|---|---|---|---|---|
| | | | Single MLP | Boosted | Gated | Mixed |
| et_h2_93 | trigram | 20k | 16.0% | 12.9% | 12.9% | 11.2 % |
| dt_s5_93 | bigram | 5k | 20.4% | 16.5% | 16.5% | 15.1% |

Table 2: Evaluation of the performance of boosting MLP acoustic models

was evaluated on a number of ARPA benchmark tests. The results are summarised in Table 2.

Initial experiments use the November 1993 Hub 2 evaluation test set (et_h2_93) . This is a 5,000 word closed vocabulary, non-verbalised punctuation test. It consists of 200 utterances, 20 from each of 10 different speakers, and is recorded using a Sennheiser HMD 410 microphone. The prompting texts are from the Wall Street Journal. Results are reported for a system using the standard ARPA 5k bigram language model.

The Spoke 5 test (dt_s5_93) is designed for evaluation of unsupervised channel adaptation algorithms. It consists of a total of 216 utterances from 10 different speakers. Each speaker's data was recorded with a different microphone. In all cases simultaneous recordings were made using a Sennheiser microphone. The task is a 5,000 word, closed vocabulary, non-verbalised punctuation test. Results are only reported for the data recorded using the Sennheiser microphone. This is a matched test since the same microphone is used to record the training data. The standard ARPA 5k bigram language model was used for the tests. Further details of the November 1993 spoke 5 and hub 2 tests, can be found in (Pallett, Fiscus, Fisher, Garofolo, Lund & Pryzbocki 1994).

Four techniques were evaluated on the WSJ corpus; a single network with 500 hidden units, a boosted ensemble with 3 networks with 500 hidden units each, a gated ensemble of the boosted networks and a mixture trained from boosted ensembles. As can be seen from the table, boosting has resulted in significant improvements in performance for both the test sets over a single model. In addition, in common with the results on the Bellcore digits, whilst the gating combination method does not give an improvement over simple averaging, the retraining of the whole ensemble using the mixed combination method gives an average improvement of a further 8% over the averaging method.

## CONCLUSION

This paper has described a number of ensemble methods for use with neural network acoustic models. It has been shown that through the use of principled methods such as boosting and mixtures the performance of these models may be improved over standard ensemble techniques. In addition, by combining the techniques via boot-strapping mixtures using the boosted networks the performance of the models can be improved further. Previous work, which focused on boosting at the word level showed improvements for a recurrent network:HMM hybrid at the word level over the baseline system (Cook & Robinson 1996). This paper has shown how the performance of a static MLP system can also be improved by boosting at the frame level.

**Acknowledgements**

Many thanks to Bellcore for providing the digits data set to our partners, ICSI; Nikki Mirghafori for help with datasets; David Johnson for providing the starting point for our code development; and Dan Kershaw for his invaluable advice.

# References

Cook, G. & Robinson, A. (1996), Boosting the performance of connectionist large-vocabulary speech recognition, *in* 'International Conference on Spoken Language Processing'.

Drucker, H., Schapire, R. & Simard, P. (1993), Improving Performance in Neural Networks Using a Boosting Algorithm, *in* S. Hanson, J. Cowan & C. Giles, eds, 'Advances in Neural Information Processing Systems 5', Morgan Kauffmann, pp. 42–49.

Hansen, L. & Salamon, P. (1990), 'Neural Network Ensembles', *IEEE Transactions on Pattern Analysis and Machine Intelligence* **12**, 993–1001.

Hochberg, M., Renals, S. & Robinson, A. (1994), 'ABBOT: The CUED hybrid connectionist-HMM large-vocabulary recognition system', *Proc. of Spoken Language Systems Technology Worshop, ARPA*.

Jacobs, R. A., Jordan, M. I., Nowlan, S. J. & Hinton, G. E. (1991), 'Adaptive mixtures of local experts', *Neural Computation* **3**(1), 79–87.

Pallett, D., Fiscus, J., Fisher, W., Garofolo, J., Lund, B. & Pryzbocki, M. (1994), '1993 Benchmark Tests for the ARPA Spoken Language Program', *ARPA Workshop on Human Language Technology* pp. 51–73. Merrill Lynch Conference Center, Plainsboro, NJ.

Paul, D. & Baker, J. (1992), The Design for the Wall Street Journal-based CSR Corpus, *in* 'DARPA Speech and Natural Language Workshop', Morgan Kaufman Publishers, Inc., pp. 357–62.

Perrone, M. P. & Cooper, L. N. (1993), When networks disagree: Ensemble methods for hybird neural networks, *in* 'Neural Networks for Speech and Image Processing', Chapman-Hall.